# Contrast adaptation in simple cells by changing the transmitter release probability

**Péter Adorján**       **Klaus Obermayer**
Dept. of Computer Science, FR2-1, Technical University Berlin
Franklinstrasse 28/29 10587 Berlin, Germany
{adp, oby} @cs.tu-berlin.de   http://www.ni.cs.tu-berlin.de

## Abstract

The contrast response function (CRF) of many neurons in the primary visual cortex saturates and shifts towards higher contrast values following prolonged presentation of high contrast visual stimuli. Using a recurrent neural network of excitatory spiking neurons with adapting synapses we show that both effects could be explained by a fast and a slow component in the synaptic adaptation. (i) Fast synaptic depression leads to saturation of the CRF and phase advance in the cortical response to high contrast stimuli. (ii) Slow adaptation of the synaptic transmitter release probability is derived such that the mutual information between the input and the output of a cortical neuron is maximal. This component—given by infomax learning rule—explains contrast adaptation of the averaged membrane potential (DC component) as well as the surprising experimental result, that the stimulus modulated component (F1 component) of a cortical cell's membrane potential adapts only weakly. Based on our results, we propose a new experiment to estimate the strength of the effective excitatory feedback to a cortical neuron, and we also suggest a relatively simple experimental test to justify our hypothesized synaptic mechanism for contrast adaptation.

## 1  Introduction

Cells in the primary visual cortex have to encode a wide range of contrast levels, and they still need to be sensitive to small changes in the input intensities. Because the signaling capacity is limited, this paradox can be resolved only by a dynamic adaptation to changes in the input intensity distribution: the contrast response function (CRF) of many neurons in the primary visual cortex shifts towards higher contrast values following prolonged presentation of high contrast visual stimuli (Ahmed et al. 1997, Carandini & Ferster 1997).

On the one hand, recent experiments, suggest that synaptic plasticity has a major role

in contrast adaptation. Because local application of GABA does not mediate adaptation (Vidyasagar 1990) and the membrane conductance does not increase significantly during adaptation (Ahmed et al. 1997, Carandini & Ferster 1997), lateral inhibition is unlikely to account for contrast adaptation. In contrast, blocking glutamate (excitatory) autoreceptors decreases the degree of adaptation (McLean & Palmer 1996). Furthermore, the adaptation is stimulus specific (e.g. Carandini et al. 1998), it is strongest if the adapting and testing stimuli are the same. On the other hand, plasticity of synaptic *weights* (e.g. Chance et al. 1998) cannot explain the weak adaptation of the stimulus driven modulations in the membrane potential (F1 component) (Carandini & Ferster 1997) and the retardation of the response phase after high contrast adaptation (Saul 1995). These experimental findings motivated us to explore how presynaptic factors, such as a long term plasticity mediated by changes in the transmitter release probability (Finlayson & Cynader 1995) affect contrast adaptation.

## 2  The single cell and the cortical circuit model

The cortical cells are modeled as leaky integrators with a firing threshold of $-55$ mV. The interspike membrane potential dynamics is described by

$$C_{\mathrm{m}}\frac{\partial V_i(t)}{\partial t} = -g_{\mathrm{leak}}\left(V_i(t) - E_{\mathrm{rest}}\right) - \sum_j g_{ij}(t)\left(V_i(t) - E_{\mathrm{syn}}\right) . \tag{1}$$

The postsynaptic conductance $g_{ij}(t)$ is the integral over the previous presynaptic events and is described by the alpha-function

$$g_{ij}(t) = \frac{g_{\max}}{\tau_{\mathrm{peak}}} \sum_s^{|\mathrm{spikes}|} p_{ij}(t_j^s) \cdot R_{ij}(t_j^s) \cdot (t - t_j^s) \cdot \exp\left(1 - \frac{t - t_j^s}{\tau_{\mathrm{peak}}}\right), \tag{2}$$

where $t_j^s$ is the arrival time of spike number $s$ from neuron $j$. Including short term synaptic depression, the effective conductance is weighted by the portion of the synaptic resource $p_{ij}(t) \cdot R_{ij}(t)$ that targets the postsynaptic side. The model parameters are $C_{\mathrm{m}} = 0.5$ nF, $g_{\mathrm{leak}} = 31$ nS, $E_{\mathrm{rest}} = -65$ mV, $E_{\mathrm{syn}} = -5$ mV, $g_{\max}^{\mathrm{exc}} = 7.8$ nS, and $\tau_{\mathrm{peak}} = 1$ ms, and the absolute refractory period is 2 ms, and after a spike, the membrane potential is reset 1 mV below the resting potential. Following Tsodyks & Markram (1997) a synapse between neurons j and i is characterized by the relative portion of the available synaptic transmitter or resource $R_{ij}$. After a presynaptic event, $R_{ij}$ decreases by $p_{ij} R_{ij}$, and recovers exponentially, where $p_{ij}$ is the transmitter release probability. The time evolution of $R_{ij}$ between two presynaptic spikes is then

$$R_{ij}(t) = 1 - (1 - (R_{ij}(t) - p_{ij}(\hat{t}) R_{ij}(\hat{t}))) \exp\left(\frac{-(t - \hat{t})}{\tau_{\mathrm{rec}}}\right) , \tag{3}$$

where $\hat{t}$ is the last spike time, and the recovery time constant $\tau_{\mathrm{rec}} = 200$ ms. Assuming Poisson distributed presynaptic firing, the steady state of the expected resource is

$$R_{ij}^{\mathrm{st}}(f_j, p_{ij}) = \frac{1}{1 + p_{ij} f_j \tau_{\mathrm{rec}}} . \tag{4}$$

The stationary mean excitatory postsynaptic current (EPSC) $I_{ij}^\infty(f_j, p_{ij})$ is proportional to the presynaptic firing frequency $f_j$ and the activated transmitter $p_{ij} R_{ij}^\infty(f_j, p_{ij})$

$$I_{ij}^\infty(f_j, p_{ij}) \propto f\, p_{ij}\, R_{ij}^\infty(f_j, p_{ij}) . \tag{5}$$

The mean current saturates for high input rates $f_j$ and it also depends on the transmitter release probability $p_{ij}$: with a high release probability the function is steeper at low presynaptic frequencies but saturates earlier than for a low release probability.

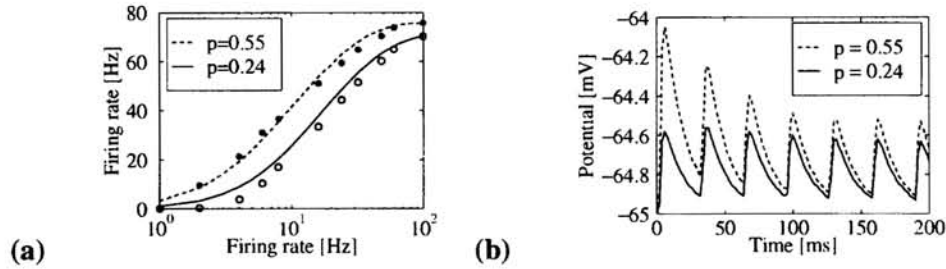

Figure 1: Short term synaptic dynamics at high and low transmitter release probability. **(a)** The estimated transfer function $O(f, p)$ for the cortical cells (Eq. 7) (solid and dashed lines) in comparison with data obtained by the integrate and fire model (Eq. 1, circles and asterisks). **(b)** EPSP trains for a series of presynaptic spikes at intervals of 31 ms (32 Hz). p=0.55 (0.24) corresponds to adaptation to 1% (50% ) contrast (see Section 4).

In order to study contrast adaptation, 30 leaky-integrator neurons are connected fully via excitatory fast adapting synapses. Each "cortical" leaky integrator neuron receives its "geniculate" input through 30 synapses. The presynaptic geniculate spike-trains are independent Poisson-processes. Modeling visual stimulation with a drifting grating, their rates are modulated sinusoidally with a temporal frequency of 2 Hz. The background activity for each individual "geniculate" source is drawn from a Gaussian distribution with a mean of 20 Hz and a standard deviation of 5 Hz. In the model the mean geniculate firing rate (Fig. 2b) and the amplitude of modulation (Fig. 2a) increases with stimulus log contrast according to the experimental data (Kaplan et al. 1987). In the following simulations CRFs are determined according to the protocol of Carandini & Ferster (1997). The CRFs are calculated using an initial adaptation period of 5 s and a subsequent series of interleaved test and re-adaptation stimuli (1 s each).

## 3   The learning rule

We propose that contrast adaptation in a visual cortical cell is a result of its goal to maximize the amount of information the cell's output conveys about the geniculate input[1]. Following (Bell & Sejnowski 1995) we derive a learning rule for the transmitter release probability $p$ to maximize the mutual information between a cortical cell's input and output. Let $O(f, p)$ be the average output firing rate, $f$ the presynaptic firing rate, and $p$ the synaptic transmitter release probability. Maximizing the mutual information is then equivalent to maximizing the entropy of a neuron's output if we assume only additive noise:

$$
\begin{aligned}
H\left[O(f,p)\right] &= -E\left[\ln \text{Prob}(O(f,p))\right] \\
&= -E\left[\ln \frac{\text{Prob}(f)}{|\partial O(f,p)/\partial f|}\right] \\
&= E\left[\ln \left|\frac{\partial O(f,p)}{\partial f}\right|\right] - E\left[\ln \text{Prob}(f)\right] . \quad (6)
\end{aligned}
$$

(In the following all equations apply locally to a synapse between neurons j and i.)

In order to derive an analytic expression for the relation between $O$ and $f$ we use the fact that the EPSP amplitude converges to its steady state relatively fast compared to the modulation of the geniculate input to the visual cortex, and that the average firing rates of the

_____
[1] A different approach of maximizing mutual information between input and output of a single spiking neuron has been developed by Stemmler & Koch (1999). For non-spiking neurons this strategy has been demonstrated experimentally by, e.g. Laughlin (1994).

presynaptic neurons are approximately similar. Thus we approximate the activation function by

$$O(f,p) \quad \propto \quad S(f)\, p\, R^{\infty}(f,p)\,, \tag{7}$$

where $S(f) = \frac{f^{\alpha}}{f+\Theta}$ accounts for the frequency dependent summation of EPSCs. The parameters $\alpha = 1.8$ and $\Theta = 15$ Hz are determined by fitting $O(f,p)$ to the firing rate of our integrate and fire single cell model (see Fig. 1a). The objective function is then maximized by a stochastic gradient ascent learning rule for the release probability $p$

$$\tau_{\text{adapt}} \frac{\partial p}{\partial t} = \frac{\partial H\,[O(f,p)]}{\partial p} = \frac{\partial}{\partial p} \ln \left| \frac{\partial O(f,p)}{\partial f} \right|\,. \tag{8}$$

Evaluating the derivatives we obtain a non-Hebbian learning rule for the transmitter release probability $p$,

$$\tau_{\text{adapt}} \frac{\partial p}{\partial t} = -2\tau_{\text{rec}} f R + \frac{1}{p} + \frac{\tau_{\text{rec}}(fa-1)}{a + \tau_{\text{rec}} p(fa-1)}\,, \tag{9}$$

where $a = \frac{\alpha}{f} - \frac{1}{f+\Theta}$, and the adaptation time constant $\tau_{\text{adapt}} = 7$ s (Ohzawa et al. 1985). This is similar in spirit to the anti-Hebbian learning mechanism for the synaptic strength proposed by Barlow & Földiák (1989) to explain adaptation phenomena. Here, the first term is proportional to the presynaptic firing rate $f$ and to the available synaptic resource $R$, suggesting a presynaptic mechanism for the learning. Because the amplitude of the EPSP is proportional to the available synaptic resource, we could interpret $R$ as an output related quantity and $-2\tau_{\text{rec}} f R$ as an anti-Hebbian learning rule for the "strength of the synapse", i.e. the probability $p$ of the transmitter release. The second term ensures that $p$ is always larger than 0. In the current model setup for the operating range of the presynaptic geniculate cells $p$ also stays always less than 1. The third term modulates the adaptation slightly and increases the release probability $p$ most if the input firing rate is close to 20 Hz, i.e. the stimulus contrast is low.

Image contrast is related to the standard deviation of the luminance levels normalized by the mean. Because ganglion cells adapt to the mean luminance, contrast adaptation in the primary visual cortex requires only the estimation of the standard deviation. In a free viewing scenario with an eye saccade frequency of 2-3 Hz, the standard deviation can be estimated based on 10-20 image samples. Thus the adaptation rate can be fast ($\tau_{\text{adapt}} = 7\,s$), and it should also be fast in order to maintain good a representation whenever visual contrast changes, e.g. by changing light conditions. Higher order moments (than the standard deviation) of the statistics of the visual world express image structure and are represented by the receptive fields' profiles. The statistics of the visual environment are relatively static, thus the receptive field profiles should be determined and constrained by another less plastic synaptic parameter, such as the maximal synaptic conductance $g_{\text{max}}$.

## 4  Results

Figure 2 shows the average geniculate input, the membrane potential, the firing rate and the response phase of the modeled cortical cells as a function of stimulus contrast. The CRFs were calculated for two adapting contrasts 1% (dashed line) and 50% (solid line). The cortical CRF saturates for high contrast stimuli (Fig. 2e). This is due to the saturation of the postsynaptic current (cf. Fig. 1a) and thus induced by the short term synaptic depression. In accordance with the experimental data (e.g. Carandini et al. 1997) the delay of the cortical response (Fig. 2f) decreases towards high contrast stimuli. This is a consequence of fast synaptic depression (c.f. Chance et al. 1998). High modulation in the input firing rate leads to a fast transient rise in the EPSC followed by a rapid depression.

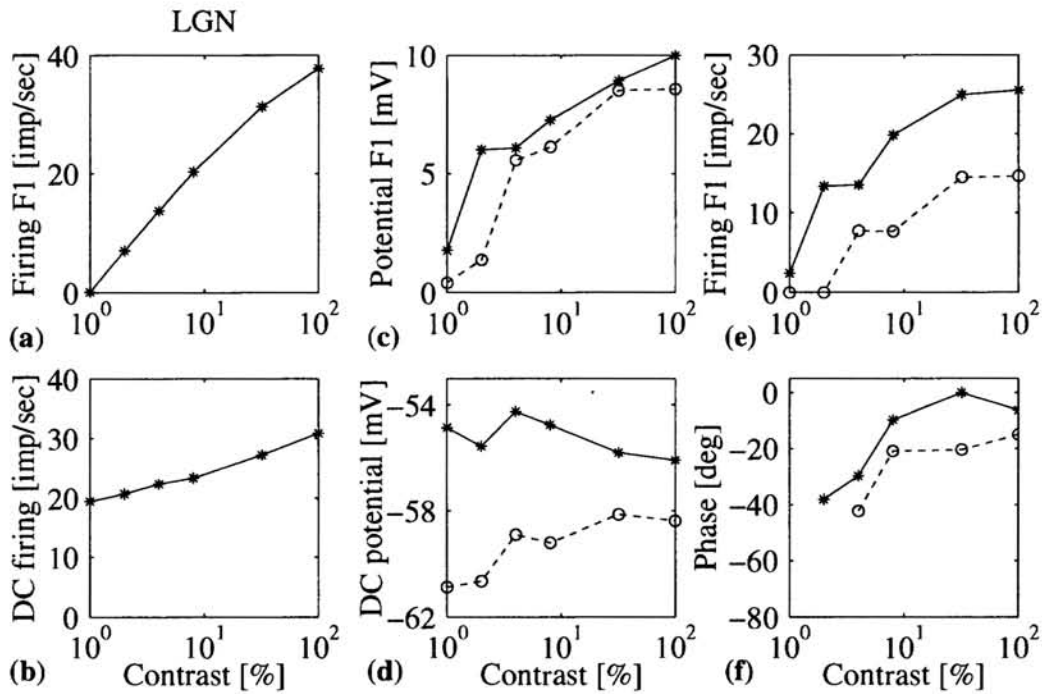

Figure 2: The DC (a) and the F1 (b) component of the geniculate input, and the response of the cortical units in the model *with* strong recurrent lateral connections and slow adaptation of the release probability on both the geniculocortical and lateral synapses. The F1 **(c)** and the DC **(d)** component of the subthreshold membrane potential of a single cortical unit, the F1 component of the firing rate **(e)**, and the response phase **(f)** are plotted as a function of stimulus contrast after adaptation to 1% (solid lines) and to 50% (dashed lines) contrast stimuli. The CRF for the membrane potential (c, d) is calculated by integrating Eq. 1 without spikes and without reset after spikes. The cortical circuitry involves strong recurrent lateral connections.

The model predicts a shift of 3-5 mV in the DC component of the subthreshold membrane potential (Fig. 2d)— a smaller amount than measured by Carandini & Ferster (1997). Nevertheless, in accordance with the data the shift caused by the adaptation is larger than the change in the DC component of the membrane potential from 1% contrast to 100% contrast. The largest shift in the DC membrane potential during adaptation occurs for small contrast stimuli because an alteration in the transmitter release probability has the largest effect on the postsynaptic current if the presynaptic firing rate is close to the geniculate background activity of 20 Hz. The maximal change in the F1 component (Fig. 2c) is around 5mV and it is half of the increase in the F1 component of the membrane potential from 1% contrast to 100% contrast. The CRF for the cortical firing rate (Fig. 2e) shifts to the right and the slope decreases after adaptation to high contrast. The model predicts that the probability $p$ for the transmitter release decreases by approximately a factor of two.

The F1 component of the cortical firing rate decreases after adaptation because after tonic decrease in the input modulated membrane potential, the over-threshold area of its F1 component decreases. The adaptation in the F1 firing rate is fed back via the recurrent excitatory connections resulting in the observable adaptation in the F1 membrane potential. Without lateral feedback (Fig. 3) the F1 component of the membrane potential is basically independent of the contrast adaptation. At high release probability a steep rise of the EPSC to a high amplitude peak is followed by rapid depression if the input is increasing. At low release probability the current increases slower to a lower amplitude, but the depression is

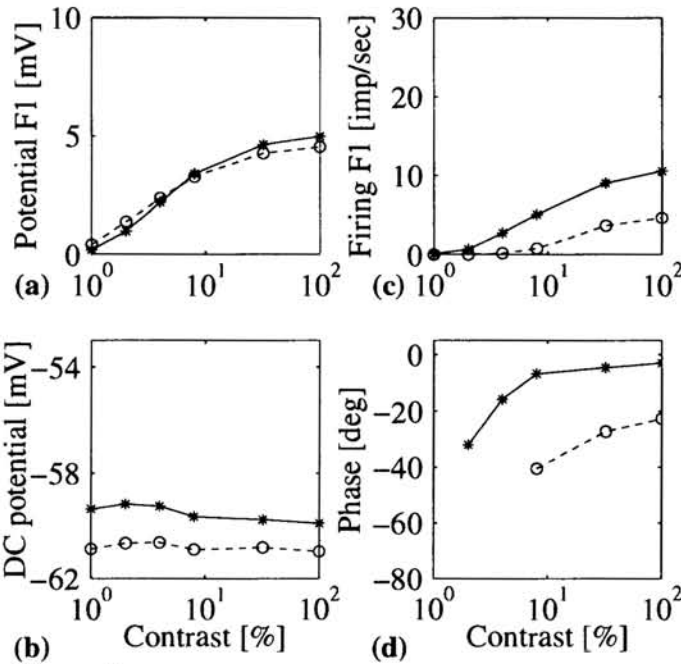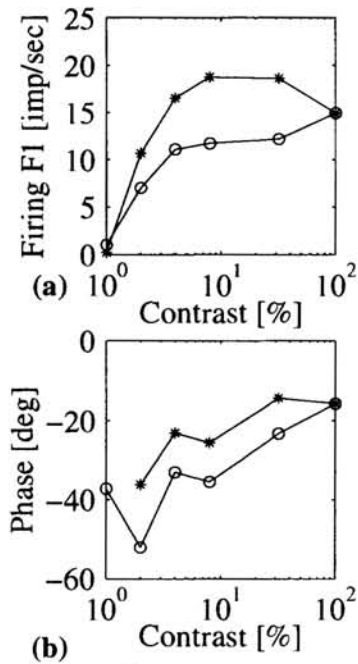

**Figure 3**  **Figure 4**

Figure 3: The membrane potential (a, b), the phase (d) of the F1 component of the firing rate, and the F1 component (c) averaged for the modeled cortical cells after adaptation to 1% (dashed lines) and 50% (solid lines) contrast. The weight of cortical connections is set to zero. The CRF for the membrane potential (a, b) is calculated by integrating Eq. 1 without spikes and without reset after spikes.

Figure 4: Hysteresis curve revealed by following the ramp method protocol (Carandini & Ferster 1997). After adaption to 1% contrast, test stimuli of 2 s duration were applied with a contrast successively increasing from 1% to 100% (asterisks), and then decreasing back to 1% (circles).

less pronounced too. As a consequence, the power at the first harmonic (F1 component) of the subthreshold membrane potential does not change if the release probability is modulated. It is modulated to a large extent by the recurrent excitatory feedback. The adaptation of the F1 component of the firing rate could therefore be used to measure the effective strength of the recurrent excitatory input to a simple cell in the primary visual cortex.

Additional simulations (data not shown) revealed that changing the transmitter release probability of the geniculocortical synapses is responsible for the adaptation in our model network. Fixing the value of $p$ for the geniculocortical synapses abolishes contrast adaptation, while fixing the release probability $p$ for the lateral synapses has no effect. Simulations show that increasing the release probability of the recurrent excitatory synapses leads to oscillatory activity (e.g. Senn et al. 1996) without altering the mean activity of simple cells. These results suggest an efficient functional segregation of feedforward and recurrent excitatory connections. Plasticity of the geniculocortical connections may play a key role in contrast adaptation, while—without affecting the CRF—plasticity of the recurrent excitatory synapses could could play a key role in dynamic feature binding and segregation in the visual cortex (e.g. Engel et al. 1997).

Figure 4 shows the averaged CRF of the cortical model neurons revealed by the ramp method (see figure caption) for strong recurrent feedback and adapting feedforward and recurrent synapses. We find hysteresis curves for the F1 component of the firing rate simi-

lar to the results reported by Carandini & Ferster (1997), and for the response phase.

In summary, by assuming two different dynamics for a single synapse we explain the saturation of the CRFs, the contrast adaptation, and the increase in the delay of the cortical response to low contrast stimuli. For the visual cortex of higher mammals, adaptation of release probability $p$ as a substrate for contrast adaptation is so far only a hypothesis. This hypothesis, however, is in agreement with the currently available data, and could additionally be justified experimentally by intracellular measurements of EPSPs evoked by stimulating the geniculocortical axons. The model predicts that after adaptation to a low contrast stimulus the amplitude of the EPSPs decreases steeply from a high value, while it shows only small changes after adaptation to a high contrast stimulus (cf. Fig. 1b).

**Acknowledgments** The authors are grateful to Christian Piepenbrock for fruitful discussions. Funded by the German Science Foundation (Ob 102/2-1, GK120-2).

## References

Ahmed, B., Allison, J. D., Douglas, R. J. & Martin, K. A. C. (1997), 'Intracellular study of the contrast-dependence of neuronal activity in cat visual cortex.', *Cerebral Cortex* **7**, 559–570.

Barlow, H. B. & Földiák, P. (1989), Adaptation and decorrelation in the cortex, *in* R. Durbin, C. Miall & C. Mitchison, eds, 'The computing neuron', Workingham: Addison-Wesley, pp. 54–72.

Bell, A. J. & Sejnowski, T. J. (1995), 'An information-maximization approach to blind sepertation and blind deconvolution', *Neur. Comput.* **7**(6), 1129–1159.

Carandini, M. & Ferster, D. (1997), 'A tonic hyperpolarization underlying contrast adaptation in cat visual cortex', *Science* **276**, 949–952.

Carandini, M., Heeger, D. J. & Movshon, J. A. (1997), 'Linearity and normalization in simple cells of the macaque primary visual cortex', *J. Neurosci.* **17**, 8621–8644.

Carandini, M., Movshon, J. A. & Ferster, D. (1998), 'Pattern adaptation and cross-orientation interactions in the primary visual cortex', *Neuropharmacology* **37**, 501–511.

Chance, F. S., Nelson, S. B. & Abbott, L. F. (1998), 'Synaptic depression and the temporal response characteristics of V1 cells', *J. Neurosci.* **18**, 4785–4799.

Engel, A. K., Roelfsema, P. R., Fries, P., Brecht, M. & Singer, W. (1997), 'Role of the temporal domain for response selection and perceptual binding', *Cerebral Cortex* **7**, 571–582.

Finlayson, P. G. & Cynader, M. S. (1995), 'Synaptic depression in visual cortex tissue slices: am in vitro model for cortical neuron adaptation', *Exp. Brain Res.* **106**, 145–155.

Kaplan, E., Purpura, K. & Shapley, R. M. (1987), 'Contrast affects the transmission of visual information through the mammalian lateral geniculate nucleus', *J. Physiol.* **391**, 267–288.

Laughlin, S. B. (1994), 'Matching coding, circuits, cells, and molecules to signals: general principles of retinal design in the fly's eye', *Prog. Ret. Eye Res.* **13**, 165–196.

McLean, J. & Palmer, L. A. (1996), 'Contrast adaptation and excitatory amino acid receptors in cat striate cortex', *Vis. Neurosci.* **13**, 1069–1087.

Ohzawa, I., Sclar, G. & Freeman, R. D. (1985), 'Contrast gain control in the cat's visual system', *J. Neurophysiol.* **54**, 651–667.

Saul, A. B. (1995), 'Adaptation in single units in visual cortex: response timing is retarted by adapting', *Vis. Neurosci.* **12**, 191–205.

Senn, W., Wyler, K., Streit, J., Larkum, M., Lüscher, H.-R., H. Mey, L. M. a. D. S., Vogt, K. & Wannier, T. (1996), 'Dynamics of a random neural network with synaptic depression', *Neural Networks* **9**, 575–588.

Stemmler, M. & Koch, C. (1999), Information maximization in single neurons, *in* 'Advances in Neural Information Processing Systems NIPS 11'. same volume.

Tsodyks, M. V. & Markram, H. (1997), 'The neural code between neocortical pyramidal neurons depends on neurotransmitter release probability', *Proc. Natl. Acad. Sci.* **94**, 719–723.

Vidyasagar, T. R. (1990), 'Pattern adaptation in cat visual cortex is a co-operative phenomenon', *Neurosci.* **36**, 175–179.
